# Discovering Structure in Continuous Variables Using Bayesian Networks

**Reimar Hofmann and Volker Tresp***
Siemens AG, Central Research
Otto-Hahn-Ring 6
81730 München, Germany

## Abstract

We study Bayesian networks for continuous variables using non-linear conditional density estimators. We demonstrate that useful structures can be extracted from a data set in a self-organized way and we present sampling techniques for belief update based on Markov blanket conditional density models.

## 1 Introduction

One of the strongest types of information that can be learned about an unknown process is the discovery of dependencies and —even more important— of independencies. A superior example is medical epidemiology where the goal is to find the causes of a disease and exclude factors which are irrelevant. Whereas complete independence between two variables in a domain might be rare in reality (which would mean that the joint probability density of variables $A$ and $B$ can be factored: $p(A, B) = p(A)p(B)$), conditional independence is more common and is often a result from true or apparent causality: consider the case that $A$ is the cause of $B$ and $B$ is the cause of $C$, then $p(C|A, B) = p(C|B)$ and $A$ and $C$ are independent under the condition that $B$ is known. Precisely this notion of cause and effect and the resulting independence between variables is represented explicitly in Bayesian networks. Pearl (1988) has convincingly argued that causal thinking leads to clear knowledge representation in form of conditional probabilities and to efficient local belief propagating rules.

Bayesian networks form a complete probabilistic model in the sense that they represent the joint probability distribution of all variables involved. Two of the powerful

---
Reimar.Hofmann@zfe.siemens.de Volker.Tresp@zfe.siemens.de

features of Bayesian networks are that any variable can be predicted from any subset of known other variables and that Bayesian networks make explicit statements about the certainty of the estimate of the state of a variable. Both aspects are particularly important for medical or fault diagnosis systems. More recently, learning of structure and of parameters in Bayesian networks has been addressed allowing for the discovery of structure between variables (Buntine, 1994, Heckerman, 1995).

Most of the research on Bayesian networks has focused on systems with discrete variables, linear Gaussian models or combinations of both. Except for linear models, continuous variables pose a problem for Bayesian networks. In Pearl's words (Pearl, 1988): "representing each [continuous] quantity by an estimated magnitude and a range of uncertainty, we quickly produce a computational mess. [Continuous variables] actually impose a computational tyranny of their own." In this paper we present approaches to applying the concept of Bayesian networks towards arbitrary nonlinear relations between continuous variables. Because they are fast learners we use Parzen windows based conditional density estimators for modeling local dependencies. We demonstrate how a parsimonious Bayesian network can be extracted out of a data set using unsupervised self-organized learning. For belief update we use local Markov blanket conditional density models which —in combination with Gibbs sampling— allow relatively efficient sampling from the conditional density of an unknown variable.

## 2  Bayesian Networks

This brief introduction of Bayesian networks follows closely Heckerman, 1995. Considering a joint probability density[1] $p(x)$ over a set of variables $\{x_1, \ldots, x_N\}$ we can decompose using the chain rule of probability

$$p(x) = \prod_{i=1}^{N} p(x_i | x_1, \ldots, x_{i-1}). \tag{1}$$

For each variable $x_i$, let the parents of $x_i$ denoted by $\mathcal{P}_i \subseteq \{x_1, \ldots, x_{i-1}\}$ be a set of variables[2] that renders $x_i$ and $\{x_1, \ldots, x_{i-1}\}$ independent, that is

$$p(x_i | x_1, \ldots, x_{i-1}) = p(x_i | \mathcal{P}_i). \tag{2}$$

Note, that $\mathcal{P}_i$ does not need to include all elements of $\{x_1, \ldots, x_{i-1}\}$ which indicates conditional independence between those variables not included in $\mathcal{P}_i$ and $x_i$ given that the variables in $\mathcal{P}_i$ are known. The dependencies between the variables are often depicted as directed acyclic[3] graphs (DAGs) with directed arcs from the members of $\mathcal{P}_i$ (the parents) to $x_i$ (the child). Bayesian networks are a natural description of dependencies between variables if they depict causal relationships between variables. Bayesian networks are commonly used as a representation of the knowledge of domain experts. Experts both define the structure of the Bayesian network and the local conditional probabilities. Recently there has been great

emphasis on learning structure and parameters in Bayesian networks (Heckerman, 1995). Most of previous work concentrated on models with only discrete variables or on linear models of continuous variables where the probability distribution of all continuous given all discrete variables is a multidimensional Gaussian. In this paper we use these ideas in context with continuous variables and nonlinear dependencies.

## 3   Learning Structure and Parameters in Nonlinear Continuous Bayesian Networks

Many of the structures developed in the neural network community can be used to model the conditional density distribution of continuous variables $p(x_i|\mathcal{P}_i)$. Under the usual signal-plus independent Gaussian noise model a feedforward neural network $NN(.)$ is a conditional density model such that $p(x_i|\mathcal{P}_i) = G(x_i; NN(\mathcal{P}_i), \sigma^2)$, where $G(x; c, \sigma^2)$ is our notation for a normal density centered at $c$ and with variance $\sigma^2$. More complex conditional densities can, for example, be modeled by mixtures of experts or by Parzen windows based density estimators which we used in our experiments (Section 5). We will use $p^M(x_i|\mathcal{P}_i)$ for a generic conditional probability model. The joint probability model is then

$$p^M(x) = \prod_{i=1}^{N} p^M(x_i|\mathcal{P}_i). \tag{3}$$

following Equations 1 and 2. Learning Bayesian networks is usually decomposed into the problems of learning structure (that is the arcs in the network) and of learning the conditional density models $P^M(x_i|\mathcal{P}_i)$ given the structure[4]. First assume the structure of the network is given. If the data set only contains complete data, we can train conditional density models $P^M(x_i|\mathcal{P}_i)$ independently of each other since the log-likelihood of the model decomposes conveniently into the individual likelihoods of the models for the conditional probabilities. Next, consider two competing network structures. We are basically faced with the well-known bias-variance dilemma: if we choose a network with too many arcs, we introduce large parameter variance and if we remove too many arcs we introduce bias. Here, the problem is even more complex since we also have the freedom to reverse arcs. In our experiments we evaluate different network structures based on the model likelihood using leave-one-out cross-validation which defines our scoring function for different network structures. More explicitly, the score for network structure $S$ is $Score = \log(p(S)) + L^{cv}$, where $p(S)$ is a prior over the network structures and $L^{cv} = \sum_{k=1}^{D} \log(p^M(x^k|S, X - \{x^k\}))$ is the leave-one-out cross-validation log-likelihood (later referred to as cv-log-likelihood). $X = \{x^k\}_{k=1}^{D}$ is the set of training samples, and $p^M(x^k|S, X - \{x^k\})$ is the probability density of sample $x_k$ given the structure $S$ and all other samples. Each of the terms $p^M(x^k|S, X - \{x^k\})$ can be computed from local densities using Equation 3.

Even for small networks it is computationally impossible to calculate the score for all possible network structures and the search for the global optimal network structure

is NP-hard. In the Section 5 we describe a heuristic search which is closely related to search strategies commonly used in discrete Bayesian networks (Heckerman, 1995).

## 4  Prior Models

In a Bayesian framework it is useful to provide means for exploiting prior knowledge, typically introducing a bias for simple structures. Biasing models towards simple structures is also useful if the model selection criteria is based on cross-validation, as in our case, because of the variance in this score. In the experiments we added a penalty per arc to the log-likelihood i.e. $\log p(S) \propto -\alpha N_A$ where $N_A$ is the number of arcs and the parameter $\alpha$ determines the weight of the penalty. Given more specific knowledge in form of a structure defined by a domain expert we can alternatively penalize the deviation in the arc structure (Heckerman, 1995). Furthermore, prior knowledge can be introduced in form of a set of artificial training data. These can be treated identical to real data and loosely correspond to the concept of a conjugate prior.

## 5  Experiment

In the experiment we used Parzen windows based conditional density estimators to model the conditional densities $p^M(x_i|\mathcal{P}_i)$ from Equation 2, i.e.

$$p^M(x_i|\mathcal{P}_i) = \frac{\sum_{k=1}^{D} G((x_i, \mathcal{P}_i); (x_i^k, \mathcal{P}_i^k), \sigma_i^2)}{\sum_{k=1}^{D} G(\mathcal{P}_i; \mathcal{P}_i^k, \sigma_i^2)}, \qquad (4)$$

where $\{x^j\}_{j=1}^{D}$ is the training set. The Gaussians in the nominator are centered at $(x_i^k, \mathcal{P}_i^k)$ which is the location of the $k$-th sample in the joint input/output (or parent/child) space and the Gaussians in the denominator are centered at $(\mathcal{P}_i^k)$ which is the location of the $k$-th sample in the input (or parent) space. For each conditional model, $\sigma_i$ was optimized using leave-one-out cross validation[5].

The unsupervised structure optimization procedure starts with a complete Bayesian model corresponding to Equation 1, i.e. a model where there is an arc between any pair of variables[6]. Next, we tentatively try all possible arc direction changes, arc removals and arc additions which do not produce directed loops and evaluate the change in score. After evaluating all legal single modifications, we accept the change which improves the score the most. The procedure stops if every arc change decreases the score. This greedy strategy can get stuck in local minima which could in principle be avoided if changes which result in worse performance are also accepted with a nonzero probability [7] (such as in annealing strategies, Heckerman, 1995). Calculating the new score at each step requires only local computation. The removal or addition of an arc corresponds to a simple removal or addition of the corresponding dimension in the Gaussians of the local density model. However,

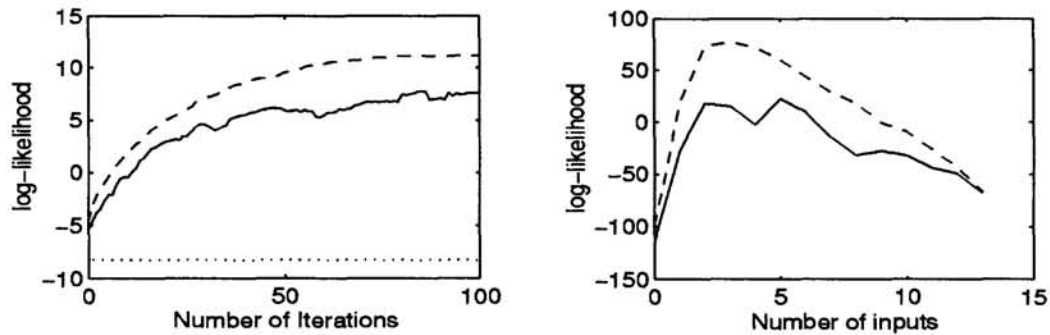

Figure 1: Left: evolution of the cv-log-likelihood (dashed) and of the log-likelihood on the test set (continuous) during structure optimization. The curves are averages over 20 runs with different partitions of training and test sets and the likelihoods are normalized with respect to the number of cv- or test-samples, respectively. The penalty per arc was $\alpha = 0.1$. The dotted line shows the Parzen joint density model commonly used in statistics, i.e. assuming no independencies and using the same width for all Gaussians in all conditional density models. Right: log-likelihood of the local conditional Parzen model for variable 3 ($p^M(x_3|\mathcal{P}_3)$) on the test set (continuous) and the corresponding cv-log-likelihood (dashed) as a function of the number of parents (inputs).

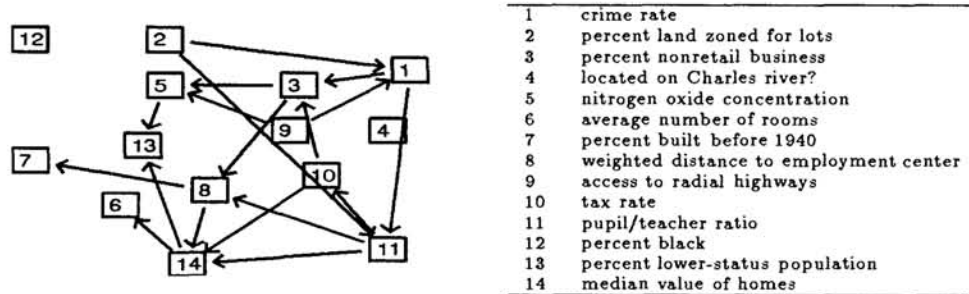

| 1 | crime rate |
|---|---|
| 2 | percent land zoned for lots |
| 3 | percent nonretail business |
| 4 | located on Charles river? |
| 5 | nitrogen oxide concentration |
| 6 | average number of rooms |
| 7 | percent built before 1940 |
| 8 | weighted distance to employment center |
| 9 | access to radial highways |
| 10 | tax rate |
| 11 | pupil/teacher ratio |
| 12 | percent black |
| 13 | percent lower-status population |
| 14 | median value of homes |

Figure 2: Final structure of a run on the full data set.

after each such operation the widths of the Gaussians $\sigma_i$ in the affected local models have to be optimized. An arc reversal is simply the execution of an arc removal followed by an arc addition.

In our experiment, we used the Boston housing data set, which contains 506 samples. Each sample consists of the housing price and 14 variables which supposedly influence the housing price in a Boston neighborhood (Figure 2). Figure 1 (left) shows an experiment where one third of the samples was reserved as a test set to monitor the process. Since the algorithm never sees the test data the increase in likelihood of the model on the test data is an unbiased estimator for how much the model has improved by the extraction of structure from the data. The large increase in the log-likelihood can be understood by studying Figure 1 (right). Here we picked a single variable (node 3) and formed a density model to predict this variable from the remaining 13 variables. Then we removed input variables in the order of their significance. After the removal of a variable, $\sigma_3$ is optimized. Note that the cv-log-likelihood increases until only three input variables are left due to the fact

that irrelevant variables or variables which are well represented by the remaining input variables are removed. The log-likelihood of the fully connected initial model is therefore low (Figure 1 left).

We did a second set of 15 runs with no test set. The scores of the final structures had a standard deviation of only 0.4. However, comparing the final structures in terms of undirected arcs[8] the difference was 18% on average. The structure from one of these runs is depicted in Figure 2 (right). In comparison to the initial complete structure with 91 arcs, only 18 arcs are left and 8 arcs have changed direction.

One of the advantages of Bayesian networks is that they can be easily interpreted. The goal of the original Boston housing data experiment was to examine whether the nitrogen oxide concentration (5) influences the housing price (14). Under the structure extracted by the algorithm, 5 and 14 are dependent given all other variables because they have a common child, 13. However, if all variables except 13 are known then they are independent. Another interesting question is what the relevant quantities are for predicting the housing price, i.e. which variables have to be known to render the housing price independent from all other variables. These are the parents, children, and children's parents of variable 14, that is variables 8, 10, 11, 6, 13 and 5. It is well known that in Bayesian networks, different constellations of directions of arcs may induce the same independencies, i.e. that the direction of arcs is not uniquely determined. It can therefore not be expected that the arcs actually reflect the direction of causality.

# 6    Missing Data and Markov Blanket Conditional Density Model

Bayesian networks are typically used in applications where variables might be missing. Given partial information (i. e. the states of a subset of the variables) the goal is to update the beliefs (i. e. the probabilities) of all unknown variables. Whereas there are powerful local update rules for networks of discrete variables without (undirected) loops, the belief update in networks with loops is in general NP-hard. A generally applicable update rule for the unknown variables in networks of discrete or continuous variables is Gibbs sampling. Gibbs sampling can be roughly described as follows: for all variables whose state is known, fix their states to the known values. For all unknown variables choose some initial states. Then pick a variable $x_i$ which is not known and update its value following the probability distribution

$$p(x_i|\{x_1, \ldots, x_N\} \setminus \{x_i\}) \propto p(x_i|\mathcal{P}_i) \prod_{x_i \in \mathcal{P}_j} p(x_j|\mathcal{P}_j). \tag{5}$$

Do this repeatedly for all unknown variables. Discard the first samples. Then, the samples which are generated are drawn from the probability distribution of the unknown variables given the known variables. Using these samples it is easy to calculate the expected value of any of the unknown variables, estimate variances, covariances and other statistical measures such as the mutual information between variables.

Gibbs sampling requires sampling from the univariate probability distribution in Equation 5 which is not straightforward in our model since the conditional density does not have a convenient form. Therefore, sampling techniques such as importance sampling have to be used. In our case they typically produce many rejected samples and are therefore inefficient. An alternative is sampling based on *Markov blanket conditional density models*. The Markov blanket of $x_i$, $\mathcal{M}_i$ is the smallest set of variables such that $p(x_i|\{x_1, \ldots, x_N\} \setminus x_i) = p(x_i|\mathcal{M}_i)$ (given a Bayesian network, the Markov blanket of a variable consists of its parents, its children and its children's parents.). The idea is to form a conditional density model $p^M(x_i|\mathcal{M}_i) \approx p(x_i|\mathcal{M}_i)$ for each variable in the network instead of computing it according to Equation 5. Sampling from this model is simple using conditional Parzen models: the conditional density is a mixture of Gaussians from which we can sample without rejection[9]. Markov blanket conditional density models are also interesting if we are only interested in always predicting one particular variable, as in most neural network applications. Assuming that a signal-plus-noise model is a reasonably good model for the conditional density, we can train an ordinary neural network to predict the variable of interest. In addition, we train a model for each input variable predicting it from the remaining variables. In addition to having obtained a model for the complete data case, we can now also handle missing inputs and do backward inference using Gibbs sampling.

## 7   Conclusions

We demonstrated that Bayesian models of local conditional density estimators form promising nonlinear dependency models for continuous variables. The conditional density models can be trained locally if training data are complete. In this paper we focused on the self-organized extraction of structure. Bayesian networks can also serve as a framework for a modular construction of large systems out of smaller conditional density models. The Bayesian framework provides consistent update rules for the probabilities i.e. communication between modules. Finally, consider input pruning or variable selection in neural networks. Note, that our pruning strategy in Figure 1 can be considered a form of variable selection by not only removing variables which are statistically independent of the output variable but also removing variables which are represented well by the remaining variables. This way we obtain more compact models. If input values are missing then the indirect influence of the pruned variables on the output will be recovered by the sampling mechanism.

## Footnotes

[1] For simplicity of notation we will only treat the continuous case. Handling mixtures of continuous and discrete variables does not impose any additional difficulties.

[2] Usually the smallest set will be used. Note that in $\mathcal{P}_i$ is defined with respect to a given ordering of the variables.

[3] i.e. not containing any *directed* loops.

[4]Differing from Heckerman we do not follow a fully Bayesian approach in which priors are defined on parameters and structure; a fully Bayesian approach is elegant if the occurring integrals can be solved in closed form which is not the case for general nonlinear models or if data are incomplete.

[5]Note that if we maintained a *global* $\sigma$ for all density estimators, we would maintain likelihood equivalence which means that each network displaying the same independence model gets the same score on any test set.

[6]The order of nodes determining the direction of initial arcs is random.

[7]In our experiments we treated very small changes in score as if they were exactly zero thus allowing small decreases in score.

[8]Since the direction of arcs is not unique we used the difference in undirected arcs to compare two structures. We used the number of arcs present in one and only one of the structures normalized with respect to the number of arcs in a fully connected network.

[9]There are, however, several open issues concerning consistency between the conditional models.

### References

Buntine, W. (1994). Operations for learning with graphical models. *Journal of Artificial Intelligence Research* 2:159-225.

Heckerman, D. (1995). A tutorial on learning Bayesian networks. Microsoft Research, TR. MSR-TR-95-06, 1995.

Pearl, J. (1988). *Probabilistic Reasoning in Intelligent Systems*. San Mateo, CA: Morgan Kaufmann.

